# Diversity Is Not All You Need: Training A Robust Cooperative Agent Needs Specialist Partners

**Rujikorn Charakorn**
VISTEC
rujikorn.c_s19@vistec.ac.th

**Poramate Manoonpong**
VISTEC, SDU
poramate.m@vistec.ac.th

**Nat Dilokthanakul**
KMITL
nat.di@kmitl.ac.th

## Abstract

Partner diversity is known to be crucial for training a robust generalist cooperative agent. In this paper, we show that partner specialization, in addition to diversity, is crucial for the robustness of a downstream generalist agent. We propose a principled method for quantifying both the diversity and specialization of a partner population based on the concept of mutual information. Then, we observe that the recently proposed cross-play minimization (`XP-min`) technique produces diverse and specialized partners. However, the generated partners are *overfit*, reducing their usefulness as training partners. To address this, we propose simple methods, based on reinforcement learning and supervised learning, for *extracting* the diverse and specialized behaviors of `XP-min` generated partners but not their overfitness. We demonstrate empirically that the proposed method effectively removes overfitness, and extracted populations produce more robust generalist agents compared to the source `XP-min` populations. This result highlights the importance of considering both the diversity and specialization of training partners while carefully managing their overfitness for training robust cooperative generalists.

## 1 Introduction

Multi-agent reinforcement learning (MARL) algorithms can generate a team of agents for solving complex cooperative tasks [1, 2]. However, these agents tend to *overfit* to teammates seen during training and cannot cooperate effectively with unseen team members [3, 4]. This problem is also known as the *ad-hoc teamwork problem* [5]. A crucial aspect of building a robust cooperative (*generalist*) agent is the diversity of training partners [6–10]. Recent state-of-the-art partner generation methods utilize *cross-play minimization* technique (`XP-min`) [11–16] to produce diverse partners.

`XP-min` approaches generate partners that are behaviorally diverse by learning incompatible policies via some measure of incompatibility, e.g., incompatible with other agents in the same population [12, 14] or best response agents [15, 16]. Fundamentally, `XP-min` aims to maximize self-play (SP) return and minimize return of cross-play (XP) trajectories, in which policies from different teams interact. However, due to the nature of `XP-min` objective, generated partners are incentivized to learn to identify the current partner and use the partner's identity to decide to cooperate (maximize return) or sabotage (minimize return). This behavior is also known as *handshaking* [17, 18]. Importantly, *handshakes*—conventions used for handshaking—could be arbitrary and nonsensical, e.g., an agent will cooperate only if the partner moves in the north direction in the first timestep. **Handshaking behaviors are undesirable** if these partner agents were to be used for training a generalist agent, simply because they overfit to random handshakes established during training. Thus, handshaking is a form of overfitness, and we use the words "overfitness" and "handshaking" interchangeably.

The root cause of handshaking induced by `XP-min` is that the partners can learn to sabotage the game if they identify that the current interaction is an XP trajectory. Recently, the problem of handshaking behaviors has been tackled by [17, 18]. The core idea of these approaches is to regularize the

diversified partners such that they still have high expected self-play (SP) return under XP trajectories, effectively *mixing* the SP and XP experiences. That is, the partners should not sabotage the game as they have to maximize the return even under XP trajectories. Therefore, the partners are no longer incentivized to learn handshakes, and these partners should behave *in good faith* regardless of which partner they interact with. We refer to this class of approaches as *mixed-play regularization* (`MP-reg`).

Sarkar et al. [18] find that it is possible for `MP-reg` to *overcorrect* the `XP-min` objective, resulting in a less diverse partner population. We hypothesize that, even with the right amount of experience mixing in `MP-reg`, the produced partners will have less *specialization*—the quality of an agent capable of solving the problem only in specific ways. We refer to this problem as *loss of specialization* (**LOS**).

We hypothesize that a generalist agent who trains with diverse but unspecialized partners will not be exposed to diverse experiences. That is because, during training, the generalist agent can pick and choose specific solutions that give higher expected returns, ignoring other potential solutions in the environment. Furthermore, the generalist agent is not required to understand the partner's intention when the partners are willing to cooperate regardless of the solution the generalist agent attempts. Thus, we posit that the *desiderata* of training partners are not only **diversity** but also **specialization**.

Our main insight is that *`XP-min` partners are not only diverse but also have rich specialized behaviors that are useful for training a generalist agent*. So, selectively using diversity and specialization of `XP-min` partners while reducing handshaking could prove valuable for training a robust cooperative agent. While `MP-reg` could increase the diversity and reduce the overfitness of generated partners, it comes with the hypothesized **LOS** problem. Hence, we are interested in an alternative method that satisfies both desiderata of good training partners. We propose a simple yet effective method to transfer the knowledge to another set of newly initialized partners via reinforcement learning (RL). Instead of reducing handshaking from `XP-min` partners via regularization, we could specifically *extract* diverse and specialized knowledge from them *after* the training process. Importantly, we aim to maintain their diversity and specialization but not the sabotaging behaviors. We call this method *specialization transfer via reinforcement learning* (`SpecTRL`). Furthermore, we present `SpecTRL DAgger`—a combination of `SpecTRL` and DAgger [19]—to stabilize the distillation process and better maintain the diversity of the source population. **Our key contributions** are as follows:

- A set of measures that quantify the quality of a training population as a whole: **diversity**, **specialization**, **overfitness** (Section 3).

- Investigation of the interaction between the proposed measures and the robustness of downstream generalist agents. We find that overfitting and lower specialization have a detrimental impact on the robustness of downstream generalist agents (Section 4).

- A novel method `SpecTRL` and `SpecTRL DAgger` that aims to transfer diversity and specialization while eliminating overfitness of `XP-min` partners (Section 5). Finally, we show that `SpecTRL DAgger` effectively reduces the number of incapable distilled partners (Section 6).

## 2 Preliminaries

We focus on cooperative environments described as decentralized partially observable Markov decision processes (Dec-POMDP, Bernstein et al. [20]). An N-player Dec-POMDP is defined by a tuple $(\mathcal{S}, \{\mathcal{A}^i\}, \{\Omega^i\}, T, O, R, \gamma, H)$, where $\mathcal{S}$ is the global **state space**. $\mathcal{A} \equiv \times^i \mathcal{A}^i$ and $\Omega \equiv \times^i \Omega^i$ are the **joint-action** and **joint-observation spaces** of all players. The probability of the next state conditioned on a state and a joint action is given by the **transition function** $T$. Players' local observations are partial views of the current state given by the **observation function** $O$. The **reward function** $R$ outputs a global reward, $\gamma$ is the **discount factor**, and $H$ is the **horizon length**.

At timestep $t$, each player observes $o_t^i$, a partial view of the global state $s_t$, and outputs an action $a_t^i \sim \pi^i(\cdot|\tau_t^i)$, where $\tau_t^i = \{o_0^i, a_0^i, r_0, ..., o_t^i\}$ is the local history of the trajectory of player $i$. Collectively, all players produce a joint action $\boldsymbol{a}_t \sim \boldsymbol{\pi} = \prod_i \pi^i$, where $\boldsymbol{\pi}$ is the joint policy. The global reward is given as $r_t = R(s_t, \boldsymbol{a}_t)$. The return of a joint trajectory $\boldsymbol{\tau} \in \mathcal{T} \equiv (\Omega \times \mathcal{A} \times \mathbb{R})^H$ is $G(\boldsymbol{\tau}) = \sum_{t=0}^{H} \gamma^t r_t$. The expected return of a joint policy $\boldsymbol{\pi}$ is $J(\boldsymbol{\pi}) = \mathbb{E}_{\boldsymbol{\tau} \sim \rho(\boldsymbol{\pi})}[G(\boldsymbol{\tau})]$, where $\rho(\boldsymbol{\pi})$ is the distribution of joint trajectories under a joint policy $\boldsymbol{\pi}$. We use bold letters to represent joint quantities. We use subscripts and superscripts to indicate different joint policies and players, respectively. We use $\boldsymbol{\pi}^{-i}$ to represent all other agents *except* $i$, i.e., $\boldsymbol{\pi}_A = \{\pi_A^i, \boldsymbol{\pi}_A^{-i}\}$. We define the expected

return of self-play (SP) trajectories as $J_{\mathrm{SP}}(\boldsymbol{\pi}_A) := J(\pi_A^i, \boldsymbol{\pi}_A^{-i})$ and the expected return of cross-play (XP) between two joint policies as $J_{\mathrm{XP}}(\boldsymbol{\pi}_A, \boldsymbol{\pi}_B) := \frac{1}{N} \sum_{i=1}^{i=N} \left[ J(\pi_A^i, \boldsymbol{\pi}_B^{-i}) + J(\pi_B^i, \boldsymbol{\pi}_A^{-i}) \right]$ and the expected return of an ad-hoc team between a generalist policy $\pi_G$ and a joint policy $\boldsymbol{\pi}$ as $J_{\mathrm{AHT}}(\pi_G, \boldsymbol{\pi}) = \frac{1}{N} \sum_{i=1}^{i=N} J(\pi_G, \boldsymbol{\pi}^{-i})$. Note that $\pi_G$ is not a joint policy and does not have any specific role assigned to it. So, it must cooperate with other agents by filling in the missing role.

We use the word *"partner"* for policies that are used for training or testing a *generalist* agent. Our main interest is training a robust generalist agent $\pi_G$ that can cooperate with unseen partners. Formally, given a population of training partners $\mathcal{P}$, the training objective of $\pi_G$ is

$$\max_{\pi_G} J_{\mathrm{AHT}}(\pi_G; \mathcal{P}) := \mathbb{E}_{\boldsymbol{\pi}_p \in \mathcal{P}} \left[ J_{\mathrm{AHT}}(\pi_G, \boldsymbol{\pi}_P) \right] \tag{1}$$

In this work, we prioritize task completion over literal task performance. Thus, we describe the robustness of $\pi_G$ as

$$\mathcal{R}(\pi_G, \mathcal{P}_{\mathrm{test}}) := \mathrm{HM}(\{\mathrm{SR}(\pi_G, \boldsymbol{\pi}_T^{-i}) \mid \boldsymbol{\pi}_T \in \mathcal{P}_{\mathrm{test}}\}), \tag{2}$$

$$\mathrm{SR}(\boldsymbol{\pi}) := \mathbb{E}_{\boldsymbol{\tau} \sim \rho(\boldsymbol{\pi})} \left[ \mathrm{S}(\boldsymbol{\tau}) \right], \tag{3}$$

where $\mathrm{HM}(\cdot)$ gives the harmonic mean of a set of scalars, $\mathrm{SR}(\boldsymbol{\pi})$ is the success rate of $\boldsymbol{\pi}$ and S, identifies whether a joint trajectory is successful.

**Cross-Play Minimization (`XP-min`):** `XP-min` technique has been recently proposed to generate diverse training partners [11–16]. Here, we describe the variant used by Charakorn et al. [12], by which the partners learn to be incompatible with others in the same population:

$$\max_{\boldsymbol{\pi}_A} J_{\texttt{XP-min}}(\boldsymbol{\pi}_A, \mathcal{P}) = \overbrace{J_{\mathrm{SP}}(\boldsymbol{\pi}_A)}^{\text{High SP return}} \overbrace{- \lambda_{\mathrm{XP}} J_{\mathrm{XP}}(\boldsymbol{\pi}_A, \boldsymbol{\pi}_+)}^{\text{Low XP return}} ; \ \forall \boldsymbol{\pi}_A \in \mathcal{P}, \tag{4}$$

$$\boldsymbol{\pi}_+ = \operatorname*{argmax}_{\boldsymbol{\pi}_+ \in (\mathcal{P} \setminus \{\boldsymbol{\pi}_A\})} J_{\mathrm{XP}}(\boldsymbol{\pi}_A, \boldsymbol{\pi}_+), \tag{5}$$

where $\boldsymbol{\pi}_+$ is the policy that is the most compatible with $\boldsymbol{\pi}_A$. In short, the `XP-min` objective optimizes a set of joint policies that are *capable* while being *incompatible* with other policies.

**Mixed-Play Regularization (`MP-reg`):** `MP-reg` [17, 18] aims to solve a fundamental problem of `XP-min`, where Eq. 4 incentivizes the partners to overfit or learn handshakes. Here, we briefly explain the `MP-reg` objective and how it helps reduce overfitness following the description of CoMeDi [18].

In addition to SP and XP episodes, `MP-reg` introduces mixed-play (MP) episodes, which have two stages: state mixing and self-play. In state mixing, the first $t'$ timesteps of an episode will evolve according to a cross-play policy, e.g., $\boldsymbol{\pi}_{\mathrm{XP}} = (\pi_A^i, \boldsymbol{\pi}_+^{-i})$ or $\boldsymbol{\pi}_{\mathrm{XP}} = (\pi_+^i, \boldsymbol{\pi}_A^{-i})$. Then, a typical SP rollout happens right after the state mixing. The training objective of `MP-reg` is to maximize the SP return starting from $s_{t'}$ produced by $\boldsymbol{\pi}_{\mathrm{XP}}$: $J_{\mathrm{MP}}(\boldsymbol{\pi}_A, \boldsymbol{\pi}_+)$. Intuitively, this objective regularizes `XP-min` agents such that they do not "sabotage" the game when interacting under XP trajectories as they still have to maximize SP return after $t'$ timesteps. The `MP-reg` objective of each $\boldsymbol{\pi}_A$ is

$$\max_{\boldsymbol{\pi}_A} J_{\texttt{MP-reg}}(\boldsymbol{\pi}_A, \mathcal{P}) = \overbrace{J_{\mathrm{SP}}(\boldsymbol{\pi}_A)}^{\text{High SP return}} \overbrace{- \lambda_{\mathrm{XP}} J_{\mathrm{XP}}(\boldsymbol{\pi}_A, \boldsymbol{\pi}_+)}^{\text{Low XP return}} \overbrace{+ J_{\mathrm{MP}}(\boldsymbol{\pi}_A, \boldsymbol{\pi}_+)}^{\text{High MP return}} ; \ \forall \boldsymbol{\pi}_A \in \mathcal{P}, \tag{6}$$

**Mutual Information (`MI`) Objective:** LIPO [12] uses both `XP-min` and `MI` to generate diverse partners. Here, we briefly describe how `MI` objective can be used in `XP-min` training [12]. A joint policy $\boldsymbol{\pi}$ is a *latent-conditioned* policy with the form $\boldsymbol{\pi}(\boldsymbol{a}|\boldsymbol{\tau}) = \mathbb{E}_{\boldsymbol{z} \sim p(\boldsymbol{z})} \prod_i \pi^i(a^i|\tau^i, z^i)$, where $\boldsymbol{z} = \{z^i\}$ is a joint latent variable and $p(\boldsymbol{z})$ is the prior distribution of $\boldsymbol{z}$. The `MI` objective is to maximize the mutual information between the observation-action pair and the latent variable of each policy: $I(\{o^i, a^i\}; z^i)$. Since the objective is intractable, we then optimize the variational lower bound of the objective:

$$I(\{o^i, a^i\}; z^i) \geq H(z^i) + \mathbb{E}_{z^i, (o^i, a^i)} [\log q_{\phi_A}(z^i | o^i, a^i)], \tag{7}$$

where $H(z^i)$ is the entropy of the random variable $z^i$, and $q_\phi$ is a neural network approximating the true posterior $p(z^i | \{o^i, a^i\})$.

## 3  Quantifying Partner Qualities

In this section, we present three measures that quantify the different qualities of a population of partners. The purpose of these measures is to allow us to compare populations and predict which ones are better at producing more robust generalists. First, we describe the characteristics of a joint trajectory and a joint policy as follows. Given a function $f : \mathcal{T} \rightarrow \mathcal{X}$, we can compute the **characteristic of a joint trajectory $\boldsymbol{\tau}$** as $x = f(\boldsymbol{\tau})$, where $\mathcal{X}$ is the *characteristic space*. For instance, $x$ could represent the frequencies of certain events under a joint trajectory $\boldsymbol{\tau}$. Consequently, we can think of the distribution of a random variable $X$ under $\boldsymbol{\pi}$, $P(X \mid \Pi = \boldsymbol{\pi})$, as the **characteristic of a joint policy $\boldsymbol{\pi}$**. That is, the probability of observing $x$, $P(X = x \mid \Pi = \boldsymbol{\pi})$, depends on the joint policy $\boldsymbol{\pi}$. This approach allows us to utilize domain knowledge through a well-crafted function $f$ for better interpretability or even learn the function when expert knowledge is unavailable. Next, we present the first two partner qualities that affect the robustness of downstream generalists based on the concept of the mutual information between $X$ and $\Pi$: **diversity** and **specialization**.

We define the **diversity** of a population of partners $\mathcal{P}$ as how *diverse* the random variable $X \in \mathcal{X}$ distributed under $\mathcal{P}$. Then, the diversity of $X$ under $\mathcal{P}$ can be calculated using the concept of entropy:

$$\boldsymbol{\mathcal{D}}(\mathcal{P}) := H(X) = - \sum_x P(x) \log P(x) = - \sum_x \mathbb{E}_{\boldsymbol{\pi}}[P(x|\boldsymbol{\pi})] \log (\mathbb{E}_{\boldsymbol{\pi}}[P(x|\boldsymbol{\pi})]), \quad (8)$$

We can see that $\boldsymbol{\mathcal{D}}(\mathcal{P})$ has direct implication to the training distribution (Eq. 1) and, consequently, the robustness of the generalist agent (Eq. 2). For instance, if the training population $\mathcal{P}_{\text{train}}$ is characteristically diverse (i.e., $\boldsymbol{\mathcal{D}}(\mathcal{P}_{\text{train}})$ is high), it is more likely that some characteristics in $\mathcal{P}_{\text{test}}$ will be covered by the training set of the generalists.

Diversity of $\mathcal{P}$ is not the only aspect that affects the robustness of a downstream generalist agent. We expect that another quality that affects the robustness of the generalist agent is the **specialization** of the training partners. We propose to measure the specialization of $\mathcal{P}$ by how *single-minded* each partner $\boldsymbol{\pi}$ is. Mathematically, we can compute the specialization of a joint policy as the negative entropy of the characteristic of that joint policy: $-H(X \mid \Pi = \boldsymbol{\pi})$. Then, we can calculate the specialization of a population by taking the average specialization of the joint policies:

$$\boldsymbol{\mathcal{S}}(\mathcal{P}) := -\mathbb{E}_{\boldsymbol{\pi}} \left[ H(X \mid \Pi = \boldsymbol{\pi}) \right] = -H(X \mid \Pi), \quad (9)$$

$$H(X \mid \Pi = \boldsymbol{\pi}) = - \sum_x P(x|\boldsymbol{\pi}) \log P(x|\boldsymbol{\pi}) \quad (10)$$

We argue that specialization directly impacts the robustness of the generalist agent based on the following rationale. For a generalist agent to effectively cooperate with specialized partners ($\boldsymbol{\mathcal{S}}(\mathcal{P}_{\text{train}})$ is high), it has to understand the partner's intention and learn various strategies because each partner is a specialist who solves the task in specific ways. In contrast, if the partners are *not* specialized ($\boldsymbol{\mathcal{S}}(\mathcal{P}_{\text{train}})$ is low), the generalist no longer needs to understand the intention of the training partner and find the easiest or the most rewarding path to complete the task.

Another crucial quality of training partners is their **overfitness**. We propose a way to quantify the overfitness of policies in a population by evaluating them against an **oracle generalist** $\pi_G^*$, which has been trained with a set of **oracle specialists** $\mathcal{P}_S^*$. The oracle specialists in $\mathcal{P}_S^*$ are assumed to be collectively diverse, individually specialized, and do not use handshakes. This means that $\pi_G^*$ is capable of solving the task in different ways thanks to the diversity and specialization of the oracle specialists. However, $\pi_G^*$ will not be able to cooperate with a partner $\boldsymbol{\pi}^{-i}$ if the partner either uses handshakes or is overfit. Thus, we can use $\pi_G^*$ as an overfitness evaluator of generated partners. Mathematically, we define the overfitness of a population as

$$\boldsymbol{\mathcal{O}}(\mathcal{P}) = 1 - R(\pi_G^*; \mathcal{P}) \quad (11)$$

Capable partners with handshaking and overfitness ($\boldsymbol{\mathcal{O}}(\mathcal{P})$ is high) are not desirable because they cannot cooperate even with the oracle generalist, which implies that they can only cooperate only if the entire team behaves under specific state-action distribution. As a result, a downstream generalist could be *underfit* because it might not discover the specific handshakes used by the partners.

## 4  Understanding Effects of Specialization and Overfitness

In this section, we aim to explore the relationship between the specialization and overfitness of a population and the robustness of downstream generalist agents. Thus, we perform a control

experiment using oracle specialists as the starting population. Then, we derive two additional populations with lower specialization and higher overfitness, respectively, while controlling the diversity of the populations to be similar to the starting population.

We use the multi-recipe Overcooked [12, 21, 22] (MIT License, Fig. 1) as the experimental platform. In short, multi-recipe Overcooked is a two-player cooperative game where agents play as chefs who aim to deliver one of six possible recipes as fast as possible. An episode terminates when a dish is delivered or the time horizon is reached. We use a handcrafted (partial) function $f$ that extracts the completed recipe of a joint trajectory.

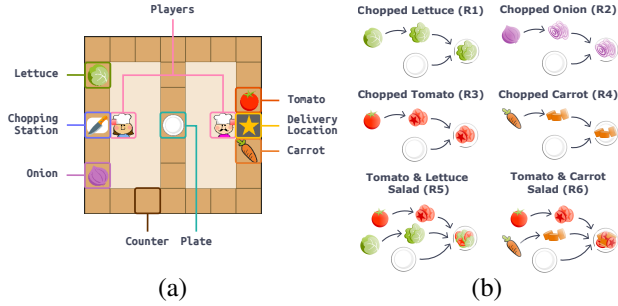

(a)          (b)

Figure 1: Overview of the multi-recipe Overcooked game.

Specifically, $x_i = f(\boldsymbol{\tau})$ is a one-hot vector representing the completed recipe in $\boldsymbol{\tau}$ and $\mathcal{X} \equiv \{x_i\}_{i=1}^{i=6} \in \mathbb{R}^6$ is the *completed recipe space*. Thus, $P(X|\Pi = \boldsymbol{\pi})$ represents the recipe completion distribution of $\boldsymbol{\pi}$ and $P(X = x_i|\boldsymbol{\pi})$ is the probability of a recipe $x_i$ being completed by the joint policy $\boldsymbol{\pi}$. We assume that these partners are capable, i.e., they have high SP returns. This means that we only consider partners that can complete the task consistently. More details can be found in Appendices A and B.

To obtain the oracle specialists, we use reward-shaped self-play training in which each specialist is trained to complete a specific recipe. We train four policies for each of the six specialist types. We use three set of specialists as **starting population** ($|\mathcal{P}_S^*| = 18$) and the other for robustness evaluation ($|\mathcal{P}_{\text{test}}| = 6$). To generate **a population with increased overfitness** ($\mathcal{P}_{\text{overfit}}^*$), we train `XP-min` agents with a modified objective. Instead of maximizing SP return, each agent learns to maximize return with a specialist. For **a population with decreased specialization** ($\mathcal{P}_{\text{unspec}}^*$), we train 18 generalist policies against $\mathcal{P}_S^*$ (see Appendix C for training details). Finally, we train 8 `XP-min` partners ($\mathcal{P}_{\texttt{XP-min}}$) using only the `XP-min` objective and put it in Table 1 and Fig. 2 for reference.

Table 1: Diversity ($\mathcal{D}$), specialization ($\mathcal{S}$), and overfitness ($\mathcal{O}$) of partner populations. The rightmost column shows the harmonic mean of success of downstream generalist agents against $\mathcal{P}_{\text{test}}$. We do not use FCP [8] in this experiment.

| Populations | $\mathcal{D}(\mathcal{P})$ | $\mathcal{S}(\mathcal{P})$ | $\mathcal{O}(\mathcal{P})$ | $\mathcal{R}(\pi_G, \mathcal{P}_{\text{test}})$ |
|---|---|---|---|---|
| (●) $\mathcal{P}_S^*$ | 1.79 | −0.01 | 0.06 | **0.81** ± 0.05 |
| (●) $\mathcal{P}_{\text{unspec}}^*$ | 1.72 | **−1.64** | 0.01 | 0.49 ± 0.07 |
| (●) $\mathcal{P}_{\text{overfit}}^*$ | 1.79 | −0.01 | **0.35** | 0.73 ± 0.01 |
| (●) $\mathcal{P}_{\texttt{XP-min}}$ | 1.31 | −0.12 | 0.51 | 0.49 ± 0.02 |

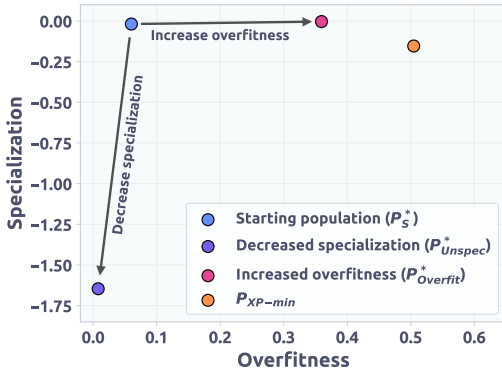

Figure 2: Relationship of training populations in the 2D specialization-overfitness landscape.

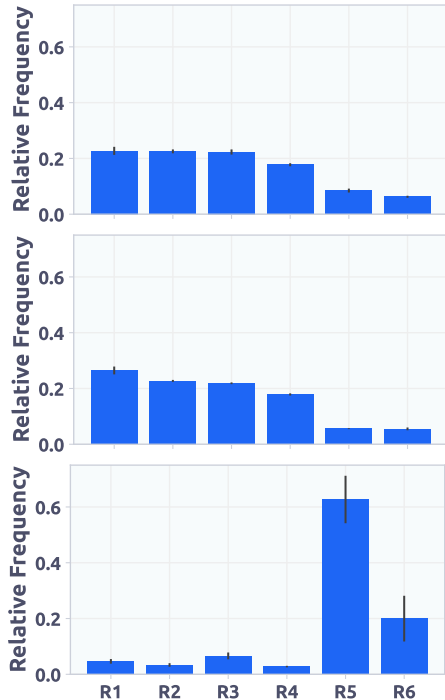

Figure 3: The recipe distribution at the last training iteration of the generalist agents trained with $\mathcal{P}_S^*$ (top), $\mathcal{P}_{\text{overfit}}^*$ (middle), and $\mathcal{P}_{\text{unspec}}^*$ (bottom).

Table 1 shows that the three control populations have comparable diversity, while $\mathcal{P}^*_{\text{overfit}}$ and $\mathcal{P}^*_{\text{unspec}}$ have increased overfitness and reduced specialization, respectively. For each population, The rightmost column shows the performance of generalist agents trained with each population when evaluated against $\mathcal{P}_{\text{test}}$. As expected, $\mathcal{P}^*_{\text{overfit}}$ and $\mathcal{P}^*_{\text{unspec}}$ produce much less robust generalists compare to $\mathcal{P}^*_S$.

We can see the root cause of the lower robustness of generalists trained with $\mathcal{P}^*_{\text{unspec}}$ in the bottom subplot of Fig. 3. We can see that the recipe distribution during the training of generalist agents is less diverse, condensing at recipe R6 (the error bars represent the standard deviations over three random seeds). Furthermore, despite learning to mostly use R5 and R6, the generalists cannot cooperate well even with specialists that prefer R5 and R6 (see Appendix D for details). In Fig. 4, we can see that the generalists *overfit* to the optimal ways for completing the most rewarding recipes (R5 and R6), achieving even higher return than self-play agents thanks to generality and flexibility of training partners in $\mathcal{P}^*_{\text{unspec}}$ (i.e., low $\mathcal{S}(\mathcal{P}^*_{\text{unspec}})$). As a result,

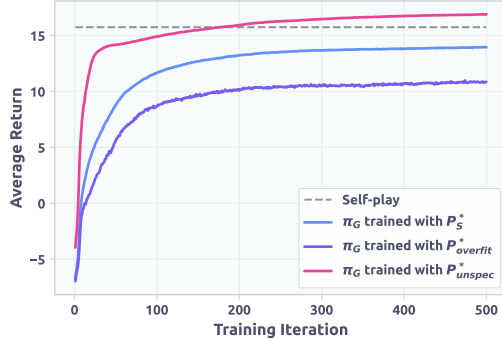

Figure 4: The average training returns of generalists trained with different oracle populations.

they are not required to have partner understanding capability, which is crucial for generalization. While $\mathcal{P}^*_{\text{overfit}}$ generate similar training recipe completion to $\mathcal{P}^*_S$, the generalist agents are not as robust as the ones trained with $\mathcal{P}^*_S$. The orange training curve in Fig. 4 suggests that the generalists trained with $\mathcal{P}^*_{\text{overfit}}$ is *underfit*. Thus, it has lower performance when matched with test partners. From these results, we conclude that **unspecialized or overfit partners are not good training partners**.

## 5 Specialization Transfer

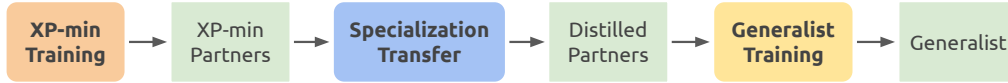

Figure 5: An overview diagram showing the steps in the training pipeline.

In Table 1, we see that $\mathcal{P}_{\texttt{XP-min}}$ has moderate diversity and high specialization but also high overfitness. This observation suggests that an `XP-min` population can potentially be a desirable training population *if the overfitness is reduced*. Therefore, we seek methods to reduce the overfitness while maintaining the diversity and specialization of an `XP-min` population. We propose `SpecTRL` and `SpecTRL DAgger` to further reduce the overfitness **after** the training process of `XP-min` agents.

`SpecTRL` distills the knowledge from a source population $\mathcal{P}$ into a distilled population $\mathcal{P}'$ with the same number of agents using reinforcement learning. Specifically, each agent $\pi_{A'} \in \mathcal{P}'$ distills the knowledge of a reference agent $\pi_A \in \mathcal{P}$ with the reward maximization objective:

$$J_{\texttt{SpecTRL}}(\boldsymbol{\pi}_{A'}) = \sum_{i=1}^{i=N} J(\pi^i_{A'}, \boldsymbol{\pi}^{-i}_A) \tag{12}$$

Intuitively, distilling via the reward maximization objective (Eq. 12) incentivizes the distilled partners to "*nudge*" the source partners to perform cooperative behaviors (which gives high return) and away from their sabotaging behaviors (which gives low return). Additionally, when the source partners cooperate, they do so in specialized ways as they have already learned specialized behaviors with `XP-min`. This means that the distilled partners must learn the preferences and specialization of the `XP-min` partners but **not** their sabotaging behaviors.

`SpecTRL` can be further combined with DAgger [19], using `XP-min` partners as experts, resulting in a knowledge transfer method that utilizes both RL and supervised learning. This combination is especially beneficial when `XP-min` partners heavily utilize complex handshakes that are unlikely to

be discovered by random exploration. We refer to this approach as `SpecTRL DAgger`:

$$J_{\text{SpecTRL DAgger}}(\boldsymbol{\pi}_{A'}) = \sum_{i=1}^{i=N} J(\pi_{A'}^i, \boldsymbol{\pi}_A^{-i}) + \lambda_{\text{DAgger}}\mathcal{L}_{\text{DAgger}}(\pi_{A'}^i), \tag{13}$$

$$\mathcal{L}_{\text{DAgger}}(\pi_{A'}^i) = -\mathbb{E}_{\tau_t^i \sim \rho(\pi_{A'}^i, \boldsymbol{\pi}_A^{-i})} \log \pi_{A'}^i(\hat{a}_t^i | \tau_t^i), \tag{14}$$

where $\hat{a}_t^i$ is the expert action, given by the source policy $\pi_A^i$, and $\lambda_{\text{DAgger}} \geq 0$. The auxiliary supervised objective Eq. 14 is useful for stabilizing the distillation process by directly transferring the knowledge from the source policy, unlike `SpecTRL` that fully relies on random exploration of RL training. A primary assumption of DAgger is that it requires access to experts' policies.

## 6 Experiments

Table 2: Qualities of oracle partner populations and their respective downstream generalists' robustness. Green arrows ($\Uparrow, \Downarrow$), red arrows ($\Downarrow$), and approximation symbol ($\approx$) indicate the improvement, degradation, and no significant changes over the source population (written between brackets), respectively. We only use the arrows when the standard deviations do not overlap. $\pm$ represents the standard deviation over three random seeds.

| Populations | $\mathcal{D}(\mathcal{P}) \uparrow$ | $\mathcal{S}(\mathcal{P}) \uparrow$ | $\mathcal{O}(\mathcal{P}) \downarrow$ | $\mathcal{R}(\pi_G, \mathcal{P}_{\text{test}}) \uparrow$ |
|---|---|---|---|---|
| *S | 1.79 | $-0.01$ | 0.06 | $0.82 \pm 0.05$ |
| *Overfit | 1.78 | 0.00 | 0.35 | $0.74 \pm 0.02$ |
| *Unspec | 1.72 | $-1.64$ | 0.01 | $0.49 \pm 0.08$ |
| [*Overfit] + `SpecTRL` | 1.79 ($\approx$) | $-0.01$ ($\approx$) | 0.16 ($\Downarrow$ **0.19**) | $0.78 \pm 0.01$ ($\Uparrow$ **0.04**) |

Table 3: Qualities of learned populations and their respective downstream generalists' robustness.

| Populations | $\mathcal{D}(\mathcal{P}) \uparrow$ | $\mathcal{S}(\mathcal{P}) \uparrow$ | $\mathcal{O}(\mathcal{P}) \downarrow$ | $\mathcal{R}(\pi_G, \mathcal{P}_{\text{test}}) \uparrow$ |
|---|---|---|---|---|
| SP | $1.08 \pm 0.05$ | $-0.49 \pm 0.12$ | $0.08 \pm 0.02$ | $0.18 \pm 0.08$ |
| `XP-min` | $1.31 \pm 0.08$ | $-0.13 \pm 0.05$ | $0.51 \pm 0.05$ | $0.56 \pm 0.07$ |
| [`XP-min`] + `MP-reg` | $1.48 \pm 0.03$ ($\Uparrow$ **0.17**) | $-0.38 \pm 0.16$ ($\Downarrow$ **0.25**) | $0.47 \pm 0.04$ ($\approx$) | $0.44 \pm 0.04$ ($\Downarrow$ **0.12**) |
| [`XP-min`] + `MI` | $1.63 \pm 0.03$ ($\Uparrow$ **0.32**) | $-0.63 \pm 0.12$ ($\Downarrow$ **0.50**) | $0.54 \pm 0.06$ ($\approx$) | $0.61 \pm 0.02$ ($\approx$) |
| [`XP-min`] + `MI` + `MP-reg` | $1.28 \pm 0.05$ ($\approx$) | $-0.68 \pm 0.21$ ($\Downarrow$ **0.55**) | $0.34 \pm 0.05$ ($\Downarrow$ **0.17**) | $0.55 \pm 0.04$ ($\approx$) |
| [`XP-min`] + `SpecTRL` | $1.21 \pm 0.10$ ($\approx$) | $-0.11 \pm 0.08$ ($\approx$) | $0.25 \pm 0.02$ ($\Downarrow$ **0.26**) | $0.58 \pm 0.08$ ($\approx$) |
| [`XP-min`] + `SpecTRL DAgger` | $1.32 \pm 0.09$ ($\approx$) | $-0.14 \pm 0.04$ ($\approx$) | $0.29 \pm 0.06$ ($\Downarrow$ **0.22**) | $0.62 \pm 0.01$ ($\approx$) |
| [`XP-min` + `MI`] + `SpecTRL` | $1.44 \pm 0.04$ ($\Downarrow$ **0.19**) | $-0.45 \pm 0.06$ ($\Uparrow$ **0.18**) | $0.20 \pm 0.06$ ($\Downarrow$ **0.34**) | $0.62 \pm 0.01$ ($\approx$) |
| [`XP-min` + `MI`] + `SpecTRL DAgger` | $1.60 \pm 0.02$ ($\approx$) | $-0.56 \pm 0.08$ ($\approx$) | $0.30 \pm 0.03$ ($\Downarrow$ **0.24**) | $0.64 \pm 0.01$ ($\Uparrow$ **0.03**) |
| [`XP-min` + `MI` + `SpecTRL DAgger`] + `SpecTRL DAgger` | $1.56 \pm 0.02$ ($\approx$) | $-0.56 \pm 0.09$ ($\approx$) | $0.27 \pm 0.02$ ($\approx$) | $0.64 \pm 0.11$ ($\approx$) |

We aim to empirically investigate how different training objectives (`XP-min`, `MI`, `MP-reg`, `SpecTRL`, and `SpecTRL DAgger`) affect the qualities of the generated partners. We use our implementation of recently proposed CoMeDi [18] to represent the `MP-reg` approach. We also include self-play (SP) as one of our baselines (see Appendix E for more details). Note that partner qualities presented in Section 3 do not depend on the size of populations of interest. In theory, we can compare populations with different sizes. For a fair comparison, we compare populations of the same size ($|\mathcal{P}| = 8$). It is possible that some generated partners are incapable due to training instability or unsuccessful

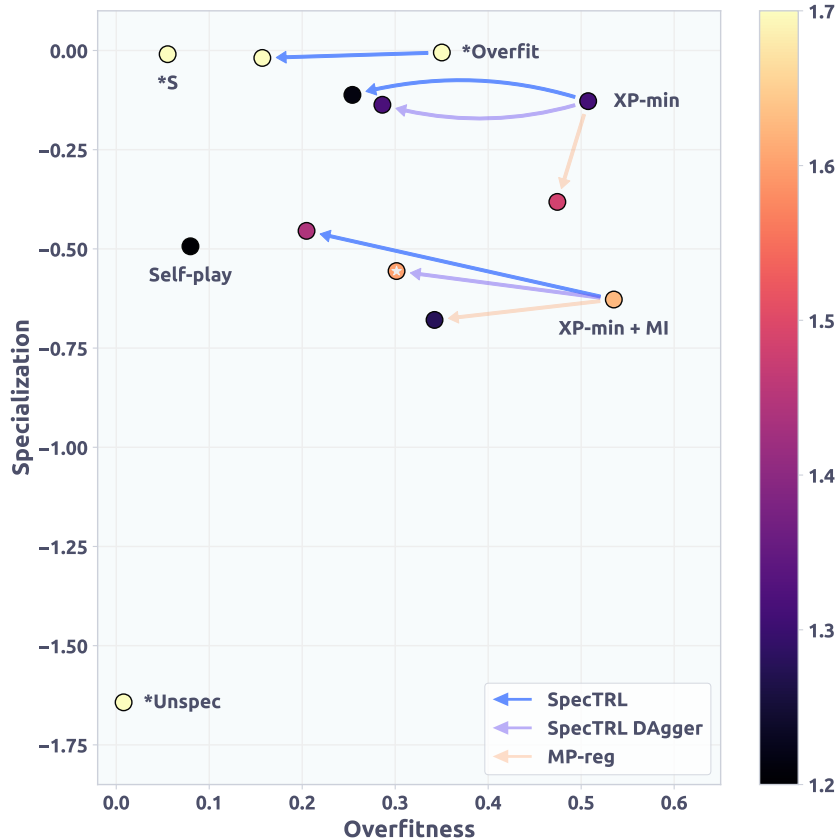

Figure 6: The partner quality landscape, representing partner qualities from Table 3 visually. The color bar represents population diversity. The arrows show how different approaches improve partners' qualities. `XP-min` + `MI` + `SpecTRL DAgger` population is marked with ⋆.

distillation. If that is the case, the proposed diversity and specialization measures are undefined for incapable partners. So, we remove those partners from diversity and specialization calculation while still keeping them in their population for training downstream generalist agents. Therefore, for a fair comparison, we always incorporate *Fictitious Co-Play* (FCP) [8] when training a generalist. FCP adds *weaker* partners to a training population, including random and incapable ones, by using past checkpoints of the partners. For populations related to oracle specialists (marked with ∗), they have a population size of 18, and we do not use FCP as they only serve as references.

Table 2 shows the effectiveness of `SpecTRL` at reducing the overfitness of the overfit oracle specialists. Table 3 shows the qualities of learned populations and the robustness of corresponding downstream generalists. We also visualize the populations in the *partner quality landscape* in Fig. 6, allowing us to visually compare populations generated by `SpecTRL`, `SpecTRL DAgger`, and `MP-reg`.

`SpecTRL` **and** `SpecTRL DAgger` **Consistently Reduce Overfitness:** `SpecTRL` consistently reduces overfitness of the input population as intended. It successfully reduces the overfitness of the overfit oracle specialists, thus improving the robustness of generalists (see Table 2). However, `SpecTRL` significantly reduces diversity when used with `XP-min` + `MI` generated populations (see Table 3). It is likely because of unsuccessful distillation, resulting in incapable distilled partners. `SpecTRL DAgger` fixes this problem effectively (see Table 4). Thus, it maintains the

Table 4: The number of capable distilled partners when using `SpecTRL` and `SpecTRL DAgger`. Other populations do not contain incapable partners.

| Populations | Capable Partners |
|---|---|
| `XP-min` | $7.33 \pm 0.47$ |
| `XP-min` + `SpecTRL` | $6.00 \pm 0.82$ |
| `XP-min` + `SpecTRL DAgger` | $7.33 \pm 0.47$ |
| `XP-min` + `MI` + `SpecTRL` | $8.00 \pm 0.00$ |
| `XP-min` + `MI` + `SpecTRL` | $5.33 \pm 1.25$ |
| `XP-min` + `MI` + `SpecTRL DAgger` | $8.00 \pm 0.00$ |

diversity of the source population while reducing overfitness. Although the reduction in overfitness is smaller than `SpecTRL`, the robustness of downstream generalists is higher, thanks to the preserved diversity. Interestingly, `SpecTRL` seems to increase the specialization of the partners while

`SpecTRL DAgger` does not. We believe that the source partners might have multi-modal behaviors, but the distilled `SpecTRL` partners might not discover all the behavioral modes via random exploration. So, the `SpecTRL` partners become more single-minded. On the other hand, `SpecTRL DAgger` does not significantly increase specialization as the DAgger objective helps transfer multi-modal behaviors directly from the source partners through the imitation learning objective.

`MP-reg` and `MI` **Increase Diversity but Lose Specialization:** We can see that `XP-min` populations are much less diverse than the ones that incorporate either `MP-reg` or `MI` regularization during the training process. This result agrees with Charakorn et al. [12] and Sarkar et al. [18] that find `MI` and `MP-reg` to help increase the diversity of `XP-min` partners. Interestingly, we find that using both `MP-reg` and `MI` simultaneously do not yield a more diverse population. However, both regularization techniques significantly reduce the specialization of the generated partners. Therefore, despite the increased diversity, the robustness of downstream generalists does not increase as the specialization is also significantly reduced. This result confirms our hypothesis that adding a regularization during `XP-min` training also comes with the loss of specialization (**LOS**) problem.

We believe that the **LOS** problem of both regularizations has different root causes. `MP-reg` incentivizes the partners to complete the task despite not being aligned with their preferences, effectively regularizing agents from having strong preferences and reducing how single-minded they are. On the other hand, `MI` induces the generated partners to have multi-modal behaviors, which could correspond to different high-level behaviors, e.g., completing different recipes.

**Repeated Distillation Does Not Reduce Overfitness Further:** So far, we can see that `SpecTRL DAgger` is effective at reducing the overfitness of `XP-min` partners. This raises the question of whether repeated distillation could further reduce the overfitness. We investigate and find that repeating distillation does not further reduce overfitness (see the last row of Table 3). We hypothesize that the sabotaging behavior has already been significantly reduced in the first round of distillation and that `SpecTRL DAgger` is effective at removing sabotaging behavior but not other kinds of overfitness, e.g., state distribution overfitness, which remains in all populations as shown in Table 3. Therefore, repeating the distillation no longer reduces the overfitness.

# 7 Discussion

Although `MP-reg` and `MI` regularization methods have the **LOS** problem, they are still necessary for increasing the diversity of the `XP-min` agents. Theoretical understanding of how regularization during `XP-min` training changes their specialization would lead to better regularization techniques that improve partner qualities, which will be crucial for building a robust cooperative agent.

The calculation for diversity (Eq. 8) and specialization (Eq. 9) depends on the probability of each $\pi$ being drawn from its population. Thus, we can alter the diversity and specialization of a population by changing how the joint policies are drawn from the population. We believe it is possible to positively modify the diversity and specialization of a population by changing how $\pi$ are drawn. We leave this investigation for future work.

The overfitness of a population $\mathcal{O}(\mathcal{P})$ presented in this paper does not separate handshakes and state-action distribution overfitness. Future work could explicitly decouple types of overfitness for further interpretability of the measure. We do not use any of the measures presented in this paper to diversify; rather, we use them as a means to understand partner qualities and their relationship to the robustness of generalist agents. Using the presented measures as diversification objectives, potentially with domain knowledge, is a worthwhile future direction for much more efficient learning algorithms. We will explore this direction in future work.

Finally, training the generalist agent with auxiliary objectives, e.g., opponent modeling, might improve robustness. However, it is unclear how such training objectives affect the relationship between the proposed measures and the downstream robustness. Understanding how these measures interact with auxiliary objectives will be critical for building robust cooperative agents.

# 8 Limitations

We note that none of the presented measures alone are representative indicators of the quality of the partners. For example, two populations could be equally diverse yet yield vastly different levels of

robustness of the generalist agent due to their difference in specialization or overfitness. There could be other qualities that impact the robustness of generalists. We will investigate this in future work.

In the experiments, we use domain knowledge for the function $f$ (i.e., how to extract the characteristic of a trajectory) and for reward shaping of the training of the oracle specialists. Both of which affect the partner quality measures. We acknowledge this as the main limitation of the experiments. Nonetheless, evaluating models without expert knowledge is challenging and is not unique to cooperative multi-agent systems. For example, evaluating LLMs requires domain knowledge to generate test scenarios or human preferences. We do not aim to automate such a notorious challenge. Instead, our proposed measures give us the option to use domain knowledge to evaluate the qualities of cooperative agents while leaving an option for learning the function $f$ as future work.

If $f$ is not well-designed, it is possible that $\sum_x P(x|\boldsymbol{\pi}) < 1$, which is not a valid probability distribution. Consequently, the diversity and specialization of the population containing $\boldsymbol{\pi}$ is undefined. For example, an untrained $\boldsymbol{\pi}$ might have an invalid $P(X|\Pi = \boldsymbol{\pi})$ if $\mathcal{X}$ is the space of reachable goals in an environment because the policy is incapable of achieving any goal.

Our experiments are performed under only a single cooperative domain, multi-recipe Overcooked. We acknowledge this limitation and aim to investigate different domains in future work. The performance of `SpecTRL` and `SpecTRL DAgger` depends on the quality of the source partner populations. Hence, the methods should not be expected to improve arbitrary partner populations.

## 9    Related Work

**Generating Diverse Partners for Training Robust Cooperative Agents:** In recent years, much efforts in the ad-hoc teamwork literature have been put into generating diverse training partners. Using domain-knowledge, one can generate diverse partners via hand-crafted policies [23–25], domain-specific reward shaping [26–28], or Quality-Diversity (QD) algorithms [29]. On the other hand, there are several techniques that can be used to generate diverse training partners without using domain knowledge including using past checkpoints [30, 8], population-based training [3, 31], a mutual information objective [32], trajectory-based diversification [33], or `XP-min` methods [11–13, 15, 17, 18]. This paper conveys an important message that there are other qualities of training partners that should be considered for training robust cooperate agents: *Diversity is not all we need*.

**Partner Qualities That Affect Robustness:** There are studies that explicitly aim to understand variables that affect the robustness of cooperative agents [10, 13]. McKee et al. [10] and Wang et al. [13] conclude that the number of training partners and diversity are critical factors for the robustness of downstream agents. Our work studies a different set of partner qualities and shows that specialization and overfitness are also crucial for training robust cooperative generalist agents. Notably, under the competitive multi-agent setting, Vinyals et al. [34] show that learning against specialized training opponents eases the learning process and increases robustness. Our work formulates the notion of *specialization* mathematically and identifies that specialization is one of the key qualities of training partners under the cooperative setting.

**Reducing `XP-min` Partners' Overfitness:** `MP-reg` methods [17, 18] aim to reduce overfitness of `XP-min` partners during their training process. The main idea is to mix SP and XP experiences such that `XP-min` partners do not learn handshaking behaviors. Unlike `MP-reg`, the proposed method reduces the overfitness of `XP-min` partners *after* the training by knowledge distillation.

**Reducing Overfitness of Neural Networks:** Using knowledge distillation for reducing overfitness is also well known in the broad machine learning literature [35–37]. Typically, the goal of knowledge distillation is to transfer knowledge of a *teacher* model to another *student* model to reduce the overfitness of the model's predictions. We use the same idea of knowledge distillation in the context of extracting diversity and specialization of `XP-min` training ("teachers") partners to another set of ("students") partners while reducing their overfitness.

## Acknowledgement

This research was supported by the Vidyasirimedhi Institute of Science and Technology (VISTEC) and Siam Commercial Bank (SCB). Additionally, it was partially funded by the Reinventing University-AI Beyond Modeling project, supported by the Ministry of Higher Education, Science, Research, and Innovation of Thailand. We thank anonymous reviewers for their thoughtful feedback.

## References

[1] Jakub Grudzien Kuba, Ruiqing Chen, Muning Wen, Ying Wen, Fanglei Sun, Jun Wang, and Yaodong Yang. Trust region policy optimisation in multi-agent reinforcement learning. In *International Conference on Learning Representations*, 2022. URL https://openreview.net/forum?id=EcGGFkNTxdJ.

[2] Muning Wen, Jakub Kuba, Runji Lin, Weinan Zhang, Ying Wen, Jun Wang, and Yaodong Yang. Multi-agent reinforcement learning is a sequence modeling problem. *Advances in Neural Information Processing Systems*, 35:16509–16521, 2022.

[3] Micah Carroll, Rohin Shah, Mark K Ho, Tom Griffiths, Sanjit Seshia, Pieter Abbeel, and Anca Dragan. On the utility of learning about humans for human-ai coordination. *Advances in neural information processing systems*, 32, 2019.

[4] Nolan Bard, Jakob N Foerster, Sarath Chandar, Neil Burch, Marc Lanctot, H Francis Song, Emilio Parisotto, Vincent Dumoulin, Subhodeep Moitra, Edward Hughes, et al. The hanabi challenge: A new frontier for ai research. *Artificial Intelligence*, 280:103216, 2020.

[5] Peter Stone, Gal A Kaminka, Sarit Kraus, and Jeffrey S Rosenschein. Ad hoc autonomous agent teams: Collaboration without pre-coordination. In *Twenty-Fourth AAAI Conference on Artificial Intelligence*, 2010.

[6] Rujikorn Charakorn, Poramate Manoonpong, and Nat Dilokthanakul. Learning to cooperate with unseen agents through meta-reinforcement learning. In *Proceedings of the 20th International Conference on Autonomous Agents and MultiAgent Systems*, pages 1478–1479, 2021.

[7] Paul Knott, Micah Carroll, Sam Devlin, Kamil Ciosek, Katja Hofmann, Anca Dragan, and Rohin Shah. Evaluating the robustness of collaborative agents. In *Proceedings of the 20th International Conference on Autonomous Agents and MultiAgent Systems*, AAMAS '21, page 1560–1562, Richland, SC, 2021. International Foundation for Autonomous Agents and Multiagent Systems. ISBN 9781450383073.

[8] DJ Strouse, Kevin McKee, Matt Botvinick, Edward Hughes, and Richard Everett. Collaborating with humans without human data. *Advances in Neural Information Processing Systems*, 34: 14502–14515, 2021.

[9] Darius Muglich, Luisa M Zintgraf, Christian A Schroeder De Witt, Shimon Whiteson, and Jakob Foerster. Generalized beliefs for cooperative ai. In *International Conference on Machine Learning*, pages 16062–16082. PMLR, 2022.

[10] Kevin R McKee, Joel Z Leibo, Charlie Beattie, and Richard Everett. Quantifying the effects of environment and population diversity in multi-agent reinforcement learning. *Autonomous Agents and Multi-Agent Systems*, 36(1):1–16, 2022.

[11] Rujikorn Charakorn, Poramate Manoonpong, and Nat Dilokthanakul. Generating diverse cooperative agents by learning incompatible policies. In *ICML 2022 Workshop AI for Agent-Based Modelling*, 2022. URL https://openreview.net/forum?id=a7vLnGKGIjY.

[12] Rujikorn Charakorn, Poramate Manoonpong, and Nat Dilokthanakul. Generating diverse cooperative agents by learning incompatible policies. In *The Eleventh International Conference on Learning Representations*, 2023. URL https://openreview.net/forum?id=UkU05GOH7_6.

[13] Xihuai Wang, Shao Zhang, Wenhao Zhang, Wentao Dong, Jingxiao Chen, Ying Wen, and Weinan Zhang. Quantifying zero-shot coordination capability with behavior preferring partners. *arXiv preprint arXiv:2310.05208*, 2023.

[14] Arrasy Rahman, Elliot Fosong, Ignacio Carlucho, and Stefano V Albrecht. Generating teammates for training robust ad hoc teamwork agents via best-response diversity. *Transactions on Machine Learning Research*, 2023. ISSN 2835-8856. URL https://openreview.net/forum?id=l5BzfQhROl.

[15] Lei Yuan, Lihe Li, Ziqian Zhang, Feng Chen, Tianyi Zhang, Cong Guan, Yang Yu, and Zhi-Hua Zhou. Learning to coordinate with anyone. In *Proceedings of the Fifth International Conference on Distributed Artificial Intelligence*, pages 1–9, 2023.

[16] Muhammad Rahman, Jiaxun Cui, and Peter Stone. Minimum coverage sets for training robust ad hoc teamwork agents. In *Proceedings of the AAAI Conference on Artificial Intelligence*, volume 38, pages 17523–17530, 2024.

[17] Brandon Cui, Andrei Lupu, Samuel Sokota, Hengyuan Hu, David J Wu, and Jakob Nicolaus Foerster. Adversarial diversity in hanabi. In *The Eleventh International Conference on Learning Representations*, 2023. URL https://openreview.net/forum?id=uLE3WF3-H_5.

[18] Bidipta Sarkar, Andy Shih, and Dorsa Sadigh. Diverse conventions for human-AI collaboration. In *Thirty-seventh Conference on Neural Information Processing Systems*, 2023. URL https://openreview.net/forum?id=MljeRycu9s.

[19] Stéphane Ross, Geoffrey Gordon, and Drew Bagnell. A reduction of imitation learning and structured prediction to no-regret online learning. In *Proceedings of the fourteenth international conference on artificial intelligence and statistics*, pages 627–635. JMLR Workshop and Conference Proceedings, 2011.

[20] Daniel S Bernstein, Robert Givan, Neil Immerman, and Shlomo Zilberstein. The complexity of decentralized control of markov decision processes. *Mathematics of operations research*, 27(4): 819–840, 2002.

[21] Sarah A. Wu, Rose E. Wang, James A. Evans, Joshua B. Tenenbaum, David C. Parkes, and Max Kleiman-Weiner. Too many cooks: Coordinating multi-agent collaboration through inverse planning. *Topics in Cognitive Science*, n/a(n/a), 2021. doi: https://doi.org/10.1111/tops.12525. URL https://onlinelibrary.wiley.com/doi/abs/10.1111/tops.12525.

[22] David Rother, Thomas Weisswange, and Jan Peters. Disentangling interaction using maximum-mentropy reinforcement learning in multi-agent systems. In *European Conference on Artificial Intelligence*, 2023.

[23] Ahana Ghosh, Sebastian Tschiatschek, Hamed Mahdavi, and Adish Singla. Towards deployment of robust cooperative ai agents: An algorithmic framework for learning adaptive policies. In *Proceedings of the 19th International Conference on Autonomous Agents and MultiAgent Systems*, pages 447–455, 2020.

[24] Annie Xie, Dylan Losey, Ryan Tolsma, Chelsea Finn, and Dorsa Sadigh. Learning latent representations to influence multi-agent interaction. In *Conference on Robot Learning*, pages 575–588. PMLR, 2021.

[25] Woodrow Zhouyuan Wang, Andy Shih, Annie Xie, and Dorsa Sadigh. Influencing towards stable multi-agent interactions. In *Conference on Robot Learning*, pages 1132–1143. PMLR, 2022.

[26] Joel Z Leibo, Edgar A Dueñez-Guzman, Alexander Vezhnevets, John P Agapiou, Peter Sunehag, Raphael Koster, Jayd Matyas, Charlie Beattie, Igor Mordatch, and Thore Graepel. Scalable evaluation of multi-agent reinforcement learning with melting pot. In *International Conference on Machine Learning*, pages 6187–6199. PMLR, 2021.

[27] Zhenggang Tang, Chao Yu, Boyuan Chen, Huazhe Xu, Xiaolong Wang, Fei Fang, Simon Du, Yu Wang, and Yi Wu. Discovering diverse multi-agent strategic behavior via reward randomization. *arXiv preprint arXiv:2103.04564*, 2021.

[28] Chao Yu, Jiaxuan Gao, Weilin Liu, Botian Xu, Hao Tang, Jiaqi Yang, Yu Wang, and Yi Wu. Learning zero-shot cooperation with humans, assuming humans are biased. In *The Eleventh International Conference on Learning Representations*, 2023. URL https://openreview.net/forum?id=TrwE8l9aJzs.

[29] Rodrigo Canaan, Xianbo Gao, Julian Togelius, Andy Nealen, and Stefan Menzel. Generating and adapting to diverse ad hoc partners in hanabi. *IEEE Transactions on Games*, 15(2):228–241, 2022.

[30] Rujikorn Charakorn, Poramate Manoonpong, and Nat Dilokthanakul. Investigating partner diversification methods in cooperative multi-agent deep reinforcement learning. In *Neural Information Processing: 27th International Conference, ICONIP 2020, Bangkok, Thailand, November 18–22, 2020, Proceedings, Part V 27*, pages 395–402. Springer, 2020.

[31] Rui Zhao, Jinming Song, Yufeng Yuan, Haifeng Hu, Yang Gao, Yi Wu, Zhongqian Sun, and Wei Yang. Maximum entropy population-based training for zero-shot human-ai coordination. In *Proceedings of the AAAI Conference on Artificial Intelligence*, volume 37, pages 6145–6153, 2023.

[32] Keane Lucas and Ross E. Allen. Any-play: An intrinsic augmentation for zero-shot coordination. In *Proceedings of the 21st International Conference on Autonomous Agents and Multiagent Systems*, AAMAS '22, page 853–861, Richland, SC, 2022. International Foundation for Autonomous Agents and Multiagent Systems. ISBN 9781450392136.

[33] Andrei Lupu, Brandon Cui, Hengyuan Hu, and Jakob Foerster. Trajectory diversity for zero-shot coordination. In *International Conference on Machine Learning*, pages 7204–7213. PMLR, 2021.

[34] Oriol Vinyals, Igor Babuschkin, Wojciech M Czarnecki, Michaël Mathieu, Andrew Dudzik, Junyoung Chung, David H Choi, Richard Powell, Timo Ewalds, Petko Georgiev, et al. Grandmaster level in starcraft ii using multi-agent reinforcement learning. *nature*, 575(7782):350–354, 2019.

[35] Cristian Buciluǎ, Rich Caruana, and Alexandru Niculescu-Mizil. Model compression. In *Proceedings of the 12th ACM SIGKDD international conference on Knowledge discovery and data mining*, pages 535–541, 2006.

[36] Geoffrey Hinton, Oriol Vinyals, and Jeff Dean. Distilling the knowledge in a neural network. *arXiv preprint arXiv:1503.02531*, 2015.

[37] Samuel Stanton, Pavel Izmailov, Polina Kirichenko, Alexander A Alemi, and Andrew G Wilson. Does knowledge distillation really work? *Advances in Neural Information Processing Systems*, 34:6906–6919, 2021.

## Reproducibility Statement

We include additional information to reproduce the experimental results in the Appendices:

- Environment details (Appendix A)
- Experimental details (Appendix B)
- Oracle-related populations (Appendix C)
- Pseudocode, implementation details, and hyperparameters (Appendix E)

The source code is available at https://anonymous.4open.science/r/dinayn-spectrl-marl/.

## Societal Impact

The paper provides additional insight into how specialization can impact the robustness of cooperative generalist agents. Also, the proposed methods aim to reduce the overfitness of training partners to better generalize generalist agents. We believe this paper will improve how future research tackles the challenges of cooperative multi-agent systems. We also hope that the proposed methods will be used to train more robust and capable generalist agents in the near future.

The choice of $f$ and $x$ can be biased, which can discriminate certain partners' behaviors in downstream uses of the partners. Finally, having a superhuman AI assistant in cooperative games might incentivize human players to actively look for AI companions instead of learning to play with other human players, which could harm their mental health and relationships in the long term.

## A   Multi-Recipe Overcooked

We use the multi-recipe version of the simplified Overcooked game from Charakorn et al. [1], which is based on the work of Wu et al. [2], Rother et al. [3] (MIT License). The game has the following cooking ingredients: lettuce, onion, tomato, and carrot. At the beginning of an episode, they are randomly placed at pre-defined positions in the game. Specifically, lettuce and onion are randomly placed on the left or the middle counter. tomato and carrot are randomly placed on the right or the middle counter. These ingredients can be used in different recipes, making each ingredient unique. Four recipes (LettuceSalad, TomatoSalad, ChoppedCarrot, ChoppedOnion) require only a single ingredient, while the other two (TomatoLettuceSalad, TomatoCarrotSalad) require two ingredients. The ingredients must be chopped at the chopping station before placing on the plate. After the required ingredients are put on the plate, they must be delivered to the delivery station.

The players have the same egocentric observation and action spaces. The observation is a set of hand-crafted features representing a local view of the environment. Specifically, we use the following features: absolute position and facing direction, relative distance to the objects and the other agent, state of the ingredients, four booleans indicating if the agent is next to a counter in four cardinal positions, held items, the state of held items, and the type and state of the items in front of the agent. These features are concatenated as a 1-D vector of length 54. At every timestep, each player has to choose one of the six possible actions: no op, move {up, down, left, right}, and interact.

An episode lasts at most 200 timesteps and terminates immediately after a successful delivery. An episode without delivery is considered unsuccessful. Consequently, $f(\boldsymbol{\tau})$ is undefined for unsuccessful trajectories. The agents are incentivized to interact with the objects and deliver as fast as possible with the following reward function:

$$r_t = r_{\text{interact}} + r_{\text{progress}} + r_{\text{complete}} - p, \tag{15}$$

where $r_{\text{interact}}$ is a shaped reward given when an agent interacts with an object for the first time in an episode, $r_{\text{progress}}$ is given when the players progress toward a recipe completion (i.e., chopping required ingredients or putting chopped ingredients on the plate), $r_{\text{complete}}$ is given upon successful delivery, and $p$ is a penalty. We use $r_{\text{interact}} = 0.5$, $r_{\text{progress}} = 1.0$, $r_{\text{complete}} = 10$, and $p = 0.1$. We note that recipes with more than one ingredient will give only slightly higher rewards ($r_{\text{interact}} + r_{\text{progress}}$) but are significantly harder to discover by random exploration than those with one ingredient.

## B  Additional Experimental Details

Since we use handcrafted characteristic function $f$ that maps a trajectory to a one-hot vector representing a completed recipe, mathematically, $f(\boldsymbol{\tau})$ is undefined for some $\boldsymbol{\tau}$ that is unsuccessful. Hence, $f$ is a partial function. From our experience, all capable agents always complete a recipe. If necessarily, we remove trajectories that the agents fail to deliver a recipe (undefined $f(\boldsymbol{\tau})$) so that $P(X|\Pi = \boldsymbol{\pi})$ is a valid probability distribution: $\sum_x P(x|\boldsymbol{\pi}) = 1$. We remove incapable agents from all quality measurements as they do not represent meaningful behaviors, though we still use them for training generalist agents. We make sure that all populations include some weaker agents by always using FCP [4] when training a generalist. We define a capable agent as an agent with more than $50\%$ success rate under SP trajectories. We estimate $P(X|\Pi = \boldsymbol{\pi})$ for each $\boldsymbol{\pi}$ using 200 SP episodes. We estimate the overfitness of each $\boldsymbol{\pi}$ using 200 ad-hoc episodes against the oracle generalist. We evaluate the robustness of each generalist using 1200 ad-hoc episodes and 200 episodes for each test specialist. Although the proposed specialization transfer approaches are not computationally expensive, they require additional computation after training the source populations. For example, a single run of `SpecTRL` or `SpecTRL DAgger` takes around 12 hours to distill a population of 8 `XP-min` agents using a desktop PC with an AMD Ryzen 9 5950X 16-Core processor with 64GB of RAM.

The details provided here are used in both Section 4 and Section 6.

## C  Oracle-Related Populations in Section 4

**Oracle Specialists** ($\mathcal{P}_S^*$)**:**    The oracle specialists are trained using self-play with a modified reward function. For each **oracle specialist**, the completion reward $r_{\text{complete}}$ is given only when completing an assigned recipe, which is unique for each specialist. We repeat the training of all six specialists three times, resulting in the oracle specialists population with size 18 ($|\mathcal{P}_S^*| = 18$).

**Overfit Oracle Specialists** ($\mathcal{P}_{\text{overfit}}^*$)**:**    We train `XP-min` agents with a modified objective. Instead of maximizing SP return, each agent learns to maximize return with a specialist while still minimizing return when matched with other `XP-min` agents in the population. Specifically, we train three LIPO populations instead of one with 18 agents because it requires lower computation and is easier to train. This is justified since the starting population contains three copies of the six specialists. The training objective for each $\boldsymbol{\pi}_A$ in $\mathcal{P}_{\text{overfit}}^*$ can be written as

$$\max_{\boldsymbol{\pi}_A} J_{\text{XP}}(\boldsymbol{\pi}_A, \boldsymbol{\pi}_{S_A}^*) - \lambda_{\text{XP}} J_{\text{XP}}(\boldsymbol{\pi}_A, \boldsymbol{\pi}_+) \; ; \forall (\boldsymbol{\pi}_A, \boldsymbol{\pi}_{S_A}) \in P, \tag{16}$$

$$P = \{(\boldsymbol{\pi}_A, \boldsymbol{\pi}_{S_A}) \mid \boldsymbol{\pi}_A \in \mathcal{P}_{\text{overfit}}^*, \boldsymbol{\pi}_{S_A}^* \in \mathcal{P}_S^*, \text{ and } A \in \{1, ..., M\}\} \tag{17}$$

$$\boldsymbol{\pi}_+ = \operatorname*{argmax}_{\boldsymbol{\pi}_+ \in (\mathcal{P}_{\text{overfit}}^* \backslash \{\boldsymbol{\pi}_A\})} J_{\text{XP}}(\boldsymbol{\pi}_A, \boldsymbol{\pi}_+) \tag{18}$$

Optimizing Eq. 16 creates a population $\mathcal{P}_{\text{overfit}}^*$ in which each partner learns to cooperate with a specific oracle specialist while incentivized to use handshakes to reduce the XP term.

**Unspecialize Oracle Specialists** ($\mathcal{P}_{\text{unspec}}^*$)**:**    We train 18 instances of generalist agents for this population. The objective for training each generalist is

$$\max_{\boldsymbol{\pi}_G} \mathbb{E}_{\boldsymbol{\pi}_S^* \in \mathcal{P}_S^*} J_{\text{ATH}}(\boldsymbol{\pi}_G, \boldsymbol{\pi}_S^*) \; ; \forall \boldsymbol{\pi}_G \in \mathcal{P}_{\text{unspec}}^* \tag{19}$$

Note that the generalists are capable of completing all recipes, meaning that they have high diversity. At the same time, they are willing to cooperate regardless of what the partner is attempting. This means that they have very low specialization. The quantity of the diversity and specialization is given in Table 1.

## D  Additional Results for The Control Experiment in Section 4

Fig. 7 shows the full evaluation performance of generalists trained with oracle-related populations generated with domain knowledge and an `XP-min` population. Interestingly, generalists trained with the unspecialized specialists $\mathcal{P}_{\text{unspec}}^*$ are less robust to unseen test partners who prefer R6 despite the training recipe distribution heavily concentrates at R5 and R6 (Fig. 3). This result suggests that unspecialized partners make downstream generalists prone to overfitting the most optimal trajectories that maximize the return. Thus, the generalist cannot effectively cooperate with the test partners.

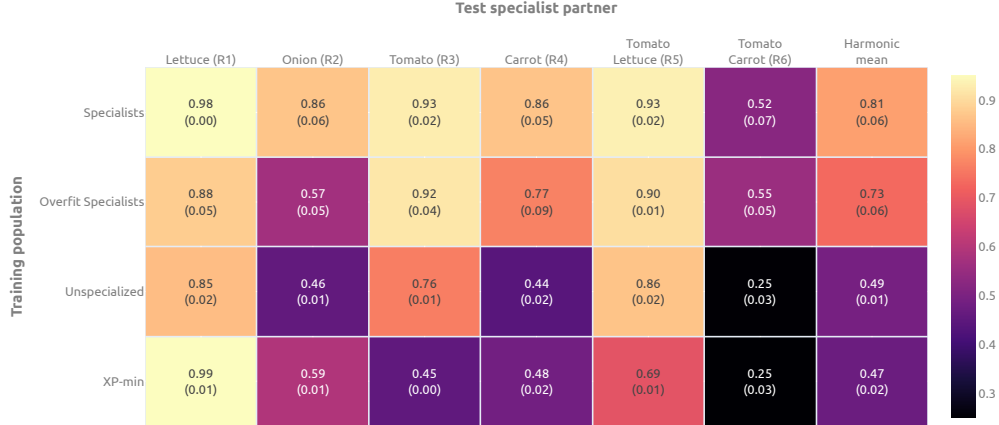

Figure 7: Test success rate of generalists trained with oracle-related populations.

# E    Implementation Details

Our implementation is based on the codebase and hyperparameters provided by Charakorn et al. [1], which uses the parameter sharing technique [5–7] and MAPPO [8] for training all partners. We provide the pseudocode for the training of the `XP-min` + `MI` + `MP-reg` here.

---

**Algorithm 1:** Training process of LIPO (on-policy)

---

This pseudocode is based on self-play. Blue text is related to the `MI` objective. `XP-min` specific code is highlighted in green. `MP-reg` related computation is in orange

**Input:** A Population $\mathcal{P} = \{\pi_A \mid 1 \le A \le N\}$

**while** *not done* **do**

    // This loop is fully parallelizable

    **for** $A \in \{1, ..., N\}$ **do**

        // Collect self-play data and compute the objective

        $\mathcal{B}_{\mathrm{SP}} \leftarrow \mathrm{CollectSPData}(\pi_A)$

        Compute $J_{\mathrm{SP}}(\pi_A)$ with $\mathcal{B}_{\mathrm{SP}}$

        // Find the most compatible policy

        $\pi_+ \leftarrow \mathrm{argmax}_{\pi_+} J_{\mathrm{XP}}(\pi_A, \pi_+)$

        // Collect cross-play data and compute the objective

        $\mathcal{B}_{\mathrm{XP}} \leftarrow \mathrm{CollectXPData}(\pi_A, \pi_+)$

        Compute $J_{\mathrm{XP}}(\pi_A, \pi_+)$ with $\mathcal{B}_{\mathrm{XP}}$

        // Collect mixed-play data and compute objective

        $\mathcal{B}_{\mathrm{MP}} \leftarrow \mathrm{CollectMPData}(\pi_A, \pi_+)$

        Compute $J_{\mathrm{MP}}(\pi_A, \pi_+)$ with $\mathcal{B}_{\mathrm{MP}}$

        // Compute mutual information objective

        Compute $L_{\mathrm{MI}}$ (the MI lower bound, Eq. 7) using all experiences ($\mathcal{B}_{\mathrm{SP}} \bigcup \mathcal{B}_{\mathrm{XP}} \bigcup \mathcal{B}_{\mathrm{MP}}$)

        // Update the policy

        $\theta_A \leftarrow \theta_A - \nabla_{\theta_A}[-J_{\mathrm{SP}}(\pi_A) + \lambda_{\mathrm{XP}} J_{\mathrm{XP}}(\pi_A, \pi_+) + \lambda_{\mathrm{MI}} L_{\mathrm{MI}} + \lambda_{\mathrm{MP}} J_{\mathrm{MP}}(\pi_A, \pi_+)]$

        // Update the MI posterior approximation

        $\phi_A \leftarrow \phi_A - \lambda_{\mathrm{MI}} \nabla_{\phi_A} L_{\mathrm{MI}}$

---

**Hyperparameters:**   We use the following hyperparameters:

- $\lambda_{\mathrm{XP}} = 0.3$ (for `XP-min` objective)
- $\lambda_{\mathrm{MI}} = 0.5$ (for `MI` lower bound objective)
- $\lambda_{\mathrm{MP}} = 0.1$ (for `MP-reg` objective)
- $\lambda_{\mathrm{DAgger}} = 0.1$ (for `SpecTRL DAgger` objective)

For populations that do not use all the training objectives, we set the corresponding coefficients to zero. We use $\lambda_{\mathrm{XP}}$ and $\lambda_{\mathrm{MI}}$ based on the values provided by Charakorn et al. [1]. We do hyperparameter search for $\lambda_{\mathrm{MP}} \in \{0.1, 0.2\}$ and $\lambda_{\mathrm{DAgger}} \in \{0.01, 0.1, 0.2\}$, and report the best result.

## F Conclusion

We propose a principled way to measure three qualities of training partners: **diversity**, **specialization**, and **overfitness**. We investigate how these qualities impact the robustness of downstream generalists and find that, in addition to diversity, specialization and overfitness are essential factors for training a robust generalist agent. We also observe that `XP-min` partners have the potential to be good training partners if their overfit is reduced. Thus, we propose two simple methods, `SpecTRL` and `SpecTRL DAgger`, to effectively reduce the overfitness while maintaining the diversity and specialization of the partners. Empirically, the proposed methods successfully reduce the overfitness of the partners. `SpecTRL DAgger` improves `SpecTRL` by stabilizing the distillation process, reducing the number of incapable distilled partners. As a result, the generalists trained with `XP-min` + `MI` + `SpecTRL DAgger` populations are the most robust. We also observe that using `MP-reg` and `MI` regularizations during `XP-min` training increases diversity. However, they have the **LOS** problem and, therefore, cannot directly increase the robustness of downstream generalists. Although the analysis done in this work relies on domain knowledge, the insight presented here is valuable for building a more robust cooperative agent in general.

## References

[1] Rujikorn Charakorn, Poramate Manoonpong, and Nat Dilokthanakul. Generating diverse cooperative agents by learning incompatible policies. In *The Eleventh International Conference on Learning Representations*, 2023. URL https://openreview.net/forum?id=UkU05GOH7_6.

[2] Sarah A. Wu, Rose E. Wang, James A. Evans, Joshua B. Tenenbaum, David C. Parkes, and Max Kleiman-Weiner. Too many cooks: Coordinating multi-agent collaboration through inverse planning. *Topics in Cognitive Science*, n/a(n/a), 2021. doi: https://doi.org/10.1111/tops.12525. URL https://onlinelibrary.wiley.com/doi/abs/10.1111/tops.12525.

[3] David Rother, Thomas Weisswange, and Jan Peters. Disentangling interaction using maximum-entropy reinforcement learning in multi-agent systems. In *European Conference on Artificial Intelligence*, 2023.

[4] DJ Strouse, Kevin McKee, Matt Botvinick, Edward Hughes, and Richard Everett. Collaborating with humans without human data. *Advances in Neural Information Processing Systems*, 34: 14502–14515, 2021.

[5] Ming Tan. Multi-agent reinforcement learning: Independent vs. cooperative agents. In *Proceedings of the tenth international conference on machine learning*, pages 330–337, 1993.

[6] Jakob Foerster, Gregory Farquhar, Triantafyllos Afouras, Nantas Nardelli, and Shimon Whiteson. Counterfactual multi-agent policy gradients. In *Proceedings of the AAAI conference on artificial intelligence*, volume 32, 2018.

[7] Tabish Rashid, Mikayel Samvelyan, Christian Schroeder, Gregory Farquhar, Jakob Foerster, and Shimon Whiteson. Qmix: Monotonic value function factorisation for deep multi-agent reinforcement learning. In *International Conference on Machine Learning*, pages 4295–4304. PMLR, 2018.

[8] Chao Yu, Akash Velu, Eugene Vinitsky, Jiaxuan Gao, Yu Wang, Alexandre Bayen, and Yi Wu. The surprising effectiveness of PPO in cooperative multi-agent games. In *Thirty-sixth Conference on Neural Information Processing Systems Datasets and Benchmarks Track*, 2022. URL https://openreview.net/forum?id=YVXaxB6L2Pl.

